# Learning Parametric Sparse Models for Image Super-Resolution

**Yongbo Li, Weisheng Dong,**[*] **Xuemei Xie, Guangming Shi**[1]**, Xin Li**[2]**, Donglai Xu**[3]
State Key Lab. of ISN, School of Electronic Engineering, Xidian University, China
[1]Key Lab. of IPIU (Chinese Ministry of Education), Xidian University, China
[2]Lane Dep. of CSEE, West Virginia University, USA
[3]Sch. of Sci. and Eng., Teesside University, UK
yongboli@stu.xidian.edu.cn, {wsdong, xmxie}@mail.xidian.edu.cn
gmshi@xidian.edu.cn, Xin.Li@mail.wvu.edu

## Abstract

Learning accurate prior knowledge of natural images is of great importance for single image super-resolution (SR). Existing SR methods either learn the prior from the low/high-resolution patch pairs or estimate the prior models from the input low-resolution (LR) image. Specifically, high-frequency details are learned in the former methods. Though effective, they are heuristic and have limitations in dealing with blurred LR images; while the latter suffers from the limitations of frequency aliasing. In this paper, we propose to combine those two lines of ideas for image super-resolution. More specifically, the parametric sparse prior of the desirable high-resolution (HR) image patches are learned from both the input low-resolution (LR) image and a training image dataset. With the learned sparse priors, the sparse codes and thus the HR image patches can be accurately recovered by solving a sparse coding problem. Experimental results show that the proposed SR method outperforms existing state-of-the-art methods in terms of both subjective and objective image qualities.

## 1  Introduction

Image super-resolution (SR) aiming to recover a high-resolution (HR) image from a single low-resolution (LR) image, has important applications in image processing and computer vision, ranging from high-definition (HD) televisions and surveillance to medical imaging. Due to the information loss in the LR image formation, image SR is a classic ill-posed inverse problem, for which strong prior knowledge of the underlying HR image is required. Generally, image SR methods can be categorized into two types, i.e., model-based and learning-based methods.

In model-based image SR, the selection of image prior is of great importance. The image priors, ranging from smoothness assumptions to sparsity and structured sparsity priors, have been exploited for image SR [1][3][4][13][14][15][19]. The smoothness prior models, e.g., Tikhonov and total variation (TV) regularizers[1], are effective in suppressing the noise but tend to over smooth image details. The sparsity-based SR methods, assuming that the HR patches have sparse representation with respect to a learned dictionary, have led to promising performances. Due to the ill-posed nature of the SR problem, designing an appropriate sparse regularizer is critical for the success of these methods. Generally, parametric sparse distributions, e.g., Laplacian and Generalized Gaussian models, which correspond to the $\ell_1$ and $\ell_p$ $(0 \leq p \leq 1)$ regularizers, are widely used. It has been shown that the SR performance can be much boosted by exploiting the structural self-similarity of natural images

---

[*]Corresponding author.

[3][4][15]. Though promising SR performance can be achieved by the sparsity-based methods, it is rather challenging to recover high-quality HR images for a large scaling factors, as there is no sufficient information for accurate estimation of the sparse models from the input LR image.

Instead of adopting a specifical prior model, learning-based SR methods learn the priors directly from a large set of LR and HR image patch pairs [2][5][6][8][18]. Specifically, mapping functions between the LR and the high-frequency details of the HR patches are learned. Popular learning-based SR methods include the sparse coding approaches[2] and the more efficient anchored neighborhood regression methods (i.e., ANR and A+)[5][6]. More recently, inspired by the great success of the deep neural network (DNN)[16] for image recognition, the DNN based SR methods have also been proposed[8], where the DNN models is used to learn the mapping functions between the LR and the high-frequency details of the HR patches. Despite the state-of-the-art performances achieved, these patch-based methods [6][8] have limitations in dealing with the blurred LR images (as shown in Sec. 5). Instead of learning high-frequency details, in [12] Li et al. proposed to learn parametric sparse distributions (i.e., non-zero mean Laplacian distributions) of the sparse codes from retrieved HR images that are similar to the LR image. State-of-the-art SR results have been achieved for the landmark LR images, for which similar HR images can be retrieved from a large image set. However, it has limitations for general LR images (i.e., it reduces to be the conventional sparsity-based SR method), for which correlated HR images cannot be found in the image database.

In this paper, we propose a novel image SR approach combining the ideas of sparsity-based and learning-based approaches for SR. The sparse prior, i.e., the parametric sparse distributions (e.g., Laplace distribution) are learned from general HR image patches. Specifically, a set of mapping functions between the LR image patches and the sparse codes of the HR patches are learned. In addition to the learned sparse prior, the learned sparse distributions are also combined with those estimated from the input LR image. Experimental results show that the proposed method performs much better than the current state-of-the-art SR approaches.

## 2    Related works

In model-based SR, it is often assumed that the desirable HR image/patches have sparse expansions with respect to a certain dictionary. For a given LR image $\boldsymbol{y} = \mathbf{H}\boldsymbol{x} + \boldsymbol{n}$, where $\mathbf{H} \in \mathbb{R}^{M \times N}$ specifies the degradation model, $\boldsymbol{x} \in \mathbb{R}^N$ and $\boldsymbol{n} \in \mathbb{R}^M$ denote the original image and additive Gaussian noise, respectively. Sparsity-based SR image reconstruction can be formulated as [3][4]

$$(\boldsymbol{x}, \boldsymbol{\alpha}) = \underset{\boldsymbol{x}, \boldsymbol{\alpha}}{\operatorname{argmin}} ||\boldsymbol{y} - \mathbf{H}\boldsymbol{x}||_2^2 + \eta \sum_i \{||\mathbf{R}_i\boldsymbol{x} - \mathbf{D}\boldsymbol{\alpha}_i||_2^2 + \lambda\psi(\boldsymbol{\alpha})\}, \qquad (1)$$

where $\mathbf{R}_i \in \mathbb{R}^{n \times N}$ denotes the matrix extracting image patch of size $\sqrt{n} \times \sqrt{n}$ at position $i$ from $\boldsymbol{x}$, $\mathbf{D} \in \mathbb{R}^{n \times K}$ denotes the dictionary that is an off-the-shelf basis or learned from an training dataset, and $\psi(\cdot)$ denotes the sparsity regularizer. As recovering $\boldsymbol{x}$ from $\boldsymbol{y}$ is an ill-posed inverse problem, the selection of $\psi(\cdot)$ is critical for the SR performance. Common selection of $\psi(\cdot)$ is the $\ell_p$-norm $(0 \leq p \leq 1)$ regularizer, where zero-mean sparse distributions of the sparse coefficients are assumed. In [12], nonzero-mean Laplacian distributions are used, leading to the following sparsity-based SR method,

$$(\boldsymbol{x}, \boldsymbol{\alpha}) = \underset{\boldsymbol{x}, \boldsymbol{\alpha}}{\operatorname{argmin}} ||\boldsymbol{y} - \mathbf{H}\boldsymbol{x}||_2^2 + \eta \sum_i \{||\mathbf{R}_i\boldsymbol{x} - \mathbf{D}\boldsymbol{\alpha}_i||_2^2 + ||\boldsymbol{\Lambda}_i(\boldsymbol{\alpha}_i - \boldsymbol{\beta}_i)||_1\}, \qquad (2)$$

where $\boldsymbol{\Lambda} = \operatorname{diag}(\frac{2\sqrt{2}\sigma_n^2}{\theta_{i,j}})$, $\boldsymbol{\theta}_i$ and $\boldsymbol{\beta}_i$ denote the standard derivation and expectation of $\boldsymbol{\alpha}_i$, respectively. It has been shown in [3] that by estimating $\{\boldsymbol{\beta}_i, \boldsymbol{\theta}_i\}$ from the nonlocal similar image patches of the input image, promising SR performance can be achieved. However, for large scaling factors, it is rather challenging to accurately estimate $\{\boldsymbol{\beta}_i, \boldsymbol{\theta}_i\}$ from the input LR image, due to the lack of sufficient information. To overcome this limitations, Li et al., propose to learn the parametric distributions from retrieved similar HR images [12] via block matching, and obtain state-of-the-art SR performance for landmark images. However, for general LR images, for which similar HR images cannot be found, the sparse prior $(\boldsymbol{\beta}_i, \boldsymbol{\theta}_i)$ cannot be learned.

Learning-based SR methods resolve the SR problem by learning mapping functions between LR and HR image patches [2][6][8]. Popular methods include the sparse coding methods [2], where LR/HR dictionary pair is jointly learned from a training set. The sparse codes of the LR patches with respect

to the LR dictionary are inferred via sparse coding and then used to reconstruct the HR patches with the HR dictionary. To reduce the computational complexity, anchored neighborhood points (ANR) and its advanced version (i.e., A+) methods [6] have been proposed. These methods first divided the patch spaces into many clusters, then LR/HR dictionary pairs are learned for each cluster. Mapping functions between the LR/HR patches are learned for each cluster via ridge regression. Recently, deep neural network (DNN) model has also been developed to learn the mapping functions between the LR and HR patches [8]. The advantages of the DNN model is that the entire SR pipeline is jointly optimized via end-to-end learning, leading to state-of-the-art SR performance. Despite the excellent performances, these learning-based methods focusing on learning the mapping functions between LR and HR patches have limitations in recovering a HR image from a blurry LR image generated by first applying a low-pass filtering followed by downsampling (as shown in Sec. 4). In this paper, we propose a novel image SR method by taking advantages of both the sparse-based and the example-based SR approaches. Specifically, mapping functions between the LR patches and the sparse codes of the desirable HR patches are learned. Hence, sparse prior can be learned from both the training patches and the input LR image. With the learned sparse prior, state-of-the-art SR performance can be achieved.

## 3 Learning Parametric Sparse Models

In this section, we first propose a novel method to learn the sparse codes of the desirable HR patches and then present the method to estimate the parametric distributions from both the predicted sparse codes and those of the LR images.

### 3.1 Learning the sparse codes from LR/HR patch pairs

For a given LR image patch $y_i \in \mathbb{R}^m$, we aim to learn the expectation of the sparse code $\alpha_i$ of the desirable HR patch $x_i$ with respect to dictionary $\mathbf{D}$. Without the loss of generality, we define the learning function as

$$\tilde{\alpha}_i = f(z_i; \mathbf{W}, b) = g(\mathbf{W} * z_i + b), \tag{3}$$

where $z_i$ denotes the feature vector extracted from the LR patch $y_i$, $\mathbf{W} \in \mathbb{R}^{K \times m}$ is the weighting matrix and $b \in \mathbb{R}^K$ is the bias, and $g(\cdot)$ denotes an activation function. Now, the remaining task is to learn the parameters of the learning function of Eq. (3). To learn the parameters, we first construct a large set of LR feature vectors and HR image patch pairs $\{(z_i, x_i)\}$, $i = 1, 2, \cdots, N$. For a given dictionary, the sparse codes $\alpha_i$ of $x_i$ can be obtained by a sparse coding algorithm. Then, the parameters $\mathcal{W} = \{\mathbf{W}, b\}$ can be learned by minimizing the following objective function

$$(\mathbf{W}, b) = \operatorname*{argmin}_{\mathbf{W}, b} \sum_{i=1}^{N} ||\alpha_i - f(z_i; \mathbf{W}, b)||_2^2. \tag{4}$$

The above optimization problem can be iteratively solved by using a stochastic gradient descent approach.

Considering the highly complexity of the mapping function between the LR feature vectors and the desirable sparse codes, we propose to learn a set of mapping functions for each possible local image structures. Specifically, the $K$-means clustering algorithm is used to cluster the LR/HR patches into $K$ clusters. Then, a mapping function is learned for each cluster. After clustering, the LR/HR patches in each cluster generally contain similar image structures, and linear mapping function would be sufficient to characterize the correlations between the LR feature vectors and the sparse codes of the desirable HR patches. Therefore, for each cluster $S_k$, the mapping function can be learned via minimizing

$$(\mathbf{W}_k, b_k) = \operatorname*{argmin}_{\mathbf{W}_k, b_k} \sum_{i \in S_k} ||\alpha_i - (\mathbf{W}_k z_i + b_k)||_2^2. \tag{5}$$

For simplicity, the bias term $b_k$ in the above equation can be absorbed into $\mathbf{W}_k$ by rewriting $\mathbf{W}_k$ and $z_i$ as $\mathbf{W}_k = [\mathbf{W}_k, b_k]$ and $z_i = [z_i^\top; 1]^\top$, respectively. Then, the parameters $\mathbf{W}_k$ can be easily solved via a least-square method.

As the HR patches in each cluster generally have similar image structures, a compact dictionary should be sufficient to represent the various HR patches. Hence, instead of learning an overcomplete dictionary for all HR patches, an orthogonal basis is learned for each cluster $S_k$. Specifically, a PCA

---
**Algorithm 1** Sparse codes learning algorithm
---
**Initialization**:

    (a) Construct a set of LR and HR image pairs $\{y, x\}$ and recover the HR images $\{\hat{x}\}$ with a conventional SR method;

    (b) Extract feature patches $z_i$, the LR and HR patches $y_i$ and $x_i$ from $\{\hat{x}, y, x\}$, respectively;

    (c) Clustering $\{z_i, y_i, x_i\}$ into $K$ clusters using $K$-means algorithm.

**Outer loop**: Iteration on $k = 1, 2, \cdots, K$

    (a) Calculate the PCA basis $\mathbf{D}_k$ for each cluster using the HR patches belong to the $k$-th cluster;

    (b) Computer the sparse codes as $\alpha_i = \mathcal{S}_\lambda(\mathbf{D}_{k_i}^\top x_i)$ for each $x_i$, $i \in S_k$;

    (c) Learn the parameters $\mathcal{W}$ of the mapping function via solving Eq. (5).
       **End for**

**Output**: $\{\mathbf{D}_k, \mathcal{W}_k\}$.

---

basis, denoted as $\mathbf{D}_k \in \mathbb{R}^{n \times n}$ is learned for each $S_k$, $k = 1, 2, \cdots, K$. Then, the sparse codes $\alpha_i$ can be easily obtained $\alpha_i = \mathcal{S}_\lambda(\mathbf{D}_{k_i}^\top x_i)$, where $\mathbf{D}_{k_i}$ denotes the PCA basis of the $k_i$-th cluster. Regarding the feature vectors $z_i$, we extract feature vectors from an initially recovered HR image, which can be obtained with a conventional sparsity-based method. Similar to [5][6], the first- and second-order gradients are extracted from the initially recovered HR image as the features. However, other more effective features can also be used. The sparse distribution learning algorithm is summarized in **Algorithm 1**.

## 3.2 Parametric sparse models estimation

After learning linearized mapping functions, denoted as $\tilde{\alpha}_i$, the estimates of $\alpha_i$ can be estimated from LR patch via Eq. (3). Based on the observation that natural images contain abundant self-repeating structures, a collection of similar patches can often be found for an exemplar patch. Then, the mean of $\alpha_i$ can be estimated as a weighted average of the sparse codes of the similar patches. As the original image is unknown, an initial estimate of the desirable HR image, denoted as $\hat{x}$ is obtained using a conventional SR method, e.g., solving Eq. (2). Then, the search of similar patches can be conducted based on $\hat{x}$. Let $\hat{x}_i$ denote the patch extracted from $\hat{x}$ at position $i$ and $\hat{x}_{i,l}$ denote the patches similar to $\hat{x}_i$ that are within the first $L$-th closest matches, $l = 1, 2, \cdots, L$. Denoted by $z_{i,l}$ the corresponding features vectors extracted from $\hat{x}$. Therefore, the mean of $\beta_i$ can be estimated by

$$\tilde{\beta}_i = \sum_{l=1}^{L} w_{i,l} \tilde{\alpha}_{i,l}, \tag{6}$$

where $w_{i,l} = \frac{1}{c} \exp(-||\hat{x}_{i,l} - \hat{x}_i||/h)$, $c$ is the normalization constant, and $h$ is the predefined parameter.

Additionally, we can also estimate the mean of space codes $\alpha_i$ directly from the intermediate estimate of target HR image. For each initially recovered HR patch $\hat{x}_i$, the sparse codes can be obtained via a sparse coding algorithm. As the patch space has been clustered into $K$ sub-spaces and a compact PCA basis is computed for each cluster, the sparse code of $\hat{x}_i$ can be easily computed as $\hat{\alpha}_{i,j} = \mathcal{S}_\lambda(\mathbf{D}_{k_i}^\top \hat{x}_{i,j})$, where $\mathcal{S}_\lambda(\cdot)$ is the soft-thresholding function with threshold $\lambda$, $k_i$ denote the cluster that $\hat{x}_i$ falls into. The sparse codes of the set of similar patches $\hat{x}_{i,l}$ can also be computed. Then, the expectation of $\beta_i$ can be estimated as

$$\hat{\beta}_i = \sum_{l=1}^{L} w_{i,j} \hat{\alpha}_{i,l}. \tag{7}$$

Then, an improved estimation of $\beta_i$ can be obtained by combining the above two estimates, i.e.,

$$\beta_i = \Delta \tilde{\beta}_i + (1 - \Delta)\hat{\beta}_i. \tag{8}$$

where $\Delta = \omega\mathrm{diag}(\delta_j) \in \mathbb{R}^{K \times K}$. Similar to [12], $\delta_j$ is set according to the energy ratio of $\tilde{\boldsymbol{\beta}}_i(j)$ and $\hat{\boldsymbol{\beta}}_i(j)$ as

$$\delta_j = \frac{r_j^2}{r_j^2 + 1/r_j^2}, \; r_j = \tilde{\boldsymbol{\beta}}_i(j)/\hat{\boldsymbol{\beta}}_i(j). \tag{9}$$

And $\omega$ is a predefined constant. After estimating $\boldsymbol{\beta}_i$, the variance of the sparse codes are estimated as

$$\boldsymbol{\theta}_i^2 = \frac{1}{L}\sum_{j=1}^{L}(\hat{\boldsymbol{\alpha}}_{i,j} - \boldsymbol{\beta}_i)^2. \tag{10}$$

The learned parametric Laplacian distributions with $\{\boldsymbol{\beta}_i, \boldsymbol{\theta}_i\}$ for image patches $\boldsymbol{x}_i$ are then used with the MAP estimator for image SR in the next section.

## 4 Image Super-Resolution with learned Parametric Sparsity Models

With the learned parametric sparse distributions $\{(\boldsymbol{\beta}_i, \boldsymbol{\theta}_i)\}$, image SR problem can be formulated as

$$(\hat{\boldsymbol{x}}, \hat{\mathbf{A}}_i) = \underset{\boldsymbol{x}_i, \mathbf{A}_i}{\arg\min} ||\boldsymbol{y} - \boldsymbol{x}\mathbf{H}||_2^2 + \eta \sum_i \{||\tilde{\mathbf{R}}_i \boldsymbol{x} - \mathbf{D}_{k_i}\mathbf{A}_i||_F^2 + \lambda \sum_{l=1}^{L} ||\boldsymbol{\Lambda}_i(\boldsymbol{\alpha}_{i,l} - \boldsymbol{\beta}_i)||_1\}, \tag{11}$$

where $\tilde{\mathbf{R}}_i\boldsymbol{x} = [\mathbf{R}_{i,1}\boldsymbol{x}, \mathbf{R}_{i,2}\boldsymbol{x}, \cdots, \mathbf{R}_{i,L}\boldsymbol{x}] \in \mathbb{R}^{n \times L}$ denotes the matrix formed by the similar patches, $\mathbf{A}_i = [\boldsymbol{\alpha}_{i,1}, \cdots, \boldsymbol{\alpha}_{i,L}]$, $\mathbf{D}_{k_i}$ denotes the selected PCA basis of the $k_i$-th cluster, and $\boldsymbol{\Lambda}_i = \mathrm{diag}(\frac{1}{\theta_{i,j}})$. In Eq. (11), the group of similar patches is assumed to follow the same estimated parametric distribution $\{\boldsymbol{\beta}_i, \boldsymbol{\theta}_i\}$. Eq. (11) can be approximately solved via alternative optimization. For fixed $\boldsymbol{x}_i$, the sets of sparse codes $\mathbf{A}_i$ can be solved by minimizing

$$\hat{\mathbf{A}}_i = \underset{\mathbf{A}_i}{\arg\min} ||\tilde{\mathbf{R}}_i\boldsymbol{x} - \mathbf{D}_{k_i}\mathbf{A}_i||_F^2 + \lambda\sum_{l=1}^{L}||\boldsymbol{\Lambda}_i(\boldsymbol{\alpha}_{i,l} - \boldsymbol{\beta}_i)||_1 \tag{12}$$

As the orthogonal PCA basis is used, the above equation can be solved in closed-form solution, i.e.,

$$\hat{\boldsymbol{\alpha}}_{i,l} = \mathcal{S}_{\boldsymbol{\tau}_i}(\mathbf{D}_{k_i}^\top \mathbf{R}_{i,l}\boldsymbol{x} - \boldsymbol{\beta}_i) + \boldsymbol{\beta}_i, \tag{13}$$

where $\boldsymbol{\tau}_i = \lambda/\boldsymbol{\theta}_i$. With estimated $\hat{\mathbf{A}}_i$, the whole image can be estimated by solving

$$\hat{\boldsymbol{x}} = \underset{\boldsymbol{x}}{\arg\min} ||\boldsymbol{y} - \boldsymbol{x}\mathbf{H}||_2^2 + \eta\sum_i ||\tilde{\mathbf{R}}_i\boldsymbol{x} - \mathbf{D}_{k_i}\mathbf{A}_i||_F^2, \tag{14}$$

which is a quadratic optimization problem and admits a closed-form solution, as

$$\hat{\boldsymbol{x}} = (\mathbf{H}^\top\mathbf{H} + \eta\sum_i \tilde{\mathbf{R}}_i^\top\tilde{\mathbf{R}}_i)^{-1}(\mathbf{H}^\top\boldsymbol{y} + \eta\sum_i \tilde{\mathbf{R}}_i^\top\mathbf{D}_{k_i}\hat{\mathbf{A}}_i), \tag{15}$$

where $\tilde{\mathbf{R}}_i^\top\tilde{\mathbf{R}}_i = \sum_{l=1}^{L}\mathbf{R}_l^\top\mathbf{R}_l$ and $\tilde{\mathbf{R}}_i^\top\mathbf{D}_{k_i}\hat{\mathbf{A}}_i = \sum_{l=1}^{L}\mathbf{R}_l^\top\mathbf{D}_{k_i}\hat{\boldsymbol{\alpha}}_{i,l}$. As the matrix to be inverted in Eq. (15) is very large, the conjugate gradient algorithm is used to compute Eq. (15). The proposed image SR algorithm is summarized in **Algorithm 2**. In **Algorithm 2**, we iteratively extract the feature patches from $\hat{\boldsymbol{x}}^{(t)}$ and learn $\tilde{\boldsymbol{\beta}}_i$ from the training set, leading to further improvements in predicting the sparse codes with the learned mapping functions.

## 5 Experimental results

In this section, we verify the performance of the proposed SR method. For fair comparisons, we use the relative small training set of images used in [2][6]. The training images are used to simulate the LR images, which are recovered by a sparsity-based method (e.g., the NCSR method [3]). Total $100,000$ features and HR patches pairs are extracted from the reconstructed HR images and the original HR images. Patches of size $7 \times 7$ are extracted from the feature images and HR images. Similar to [5][6], the PCA technique is used to reduce the dimensions of the feature vectors. The training patches are clustered into 1000 clusters. The other major parameters of the proposed SR

---
**Algorithm 2** Image SR with Learned Sparse Representation
---
**Initialization**:

    (a) Initialize $\hat{\boldsymbol{x}}^{(0)}$ with a conventional SR method;

    (b) Set parameters $\eta$ and $\lambda$;

**Outer loop**: Iteration over $t = 0, 1, \cdots, T$

    (a) Extract feature vectors $\boldsymbol{z}_i$ from $\hat{\boldsymbol{x}}^{(t)}$ and cluster the patches into clusters;

    (b) Learn $\tilde{\boldsymbol{\beta}}_i$ for each local patch using Eq. (6);

    (c) Update the estimate of $\boldsymbol{\beta}_i$ using Eq. (8) and estimate $\boldsymbol{\theta}_i$ with Eq. (10);

    (d) **Inner loop** (solve Eq.(11)): iteration over $j = 1, 2, \cdots, J$;
        (I) Compute $\mathbf{A}_i^{(j+1)}$ by solving Eq.(13);
        (II) Update the whole image $\hat{\boldsymbol{x}}^{(j+1)}$ via Eq. (15);
        (III) Set $\boldsymbol{x}^{(t+1)} = \boldsymbol{x}^{(j+1)}$ if $j = J$.
        **End for**

**Output**: $\boldsymbol{x}^{(t+1)}$.

---

method are set as: $L = 12$, $T = 8$, and $J = 10$. The proposed SR method is compared with several current state-of-the-art image SR methods, i.e., the sparse coding based SR method (denoted as SCSR)[2], the SR method based on sparse regression and natural image prior (denoted as KK) [7], the A+ method [6], the recent SRCNN method [8], and the NCSR method [3]. Note that the NCSR is the current sparsity-based SR method. Three images sets, i.e., Set5[9], Set14[10] and BSD100[11], which consists of 5, 14 and 100 images respectively, are used as the test images.

In this paper, we consider two types of degradation when generating the LR images, i.e., the bicubic image resizing function implemented with *imresize* in matlab and Gaussian blurring followed by downsampling with a scaling factor, both of which are commonly used in the literature of image SR.

## 5.1 Image SR for LR images generated with bicubic interpolation function

In [2][6][7][8], the LR images are generated with the bicubic interpolation function (i.e., *imresize* function in Matlab), i.e., $\boldsymbol{y} = \mathcal{B}(\boldsymbol{x}) + \boldsymbol{n}$, where $\mathcal{B}(\cdot)$ denotes the bicubic downsampling function. To deal with this type of degradation, we implement the degradation matrix $\mathbf{H}$ as an operator that resizes a HR image using bicubic function with scaling factors of $\frac{1}{s}$ and implement $\mathbf{H}^\top$ as an operator that upscales a LR image using bicubic function with scaling factor $s$, where $s = 2, 3, 4$. The average PSNR and SSIM results of the reconstructed HR images are reported in Table 1. It can be seen that the SRCNN method performs better than the A+ and the SCSR methods. It is surprising to see that the NCSR method, which only exploits the internal similar samples performs comparable with the SRCNN method. By exploiting both the external image patches and the internal similar patches, the proposed method outperforms the NCSR. The average PSNR gain over SRCNN can be up to $0.64$ dB. Parts of some reconstructed HR images by the test methods are shown in Fig. 1, from which we can see that the proposed method reproduces the most visually pleasant HR images than other competing methods. Please refer to the supplementary file for more visual comparison results.

## 5.2 Image SR for LR images generated with Gaussian blur followed by downsampling

Another commonly used degradation process is to first apply a Gaussian kernel followed by downsampling. In this experimental setting, the $7 \times 7$ Gaussian kernel of standard deviation of 1.6 is used, followed by downsampling with scaling factor $s = 2, 3, 4$. For these SCSR, KK, A+ and SRCNN methods, which cannot deal with the Gaussian blur kernel, the iterative back-projection [17] method is applied to the reconstructed HR images by those methods as a post processing to remove the blur. The average PSNR and SSIM results on the three test image sets are reported in Table 2. It can be seen that the performance of the example-based methods, i.e., SCSR[2], KK[7], A+[6] and SRCNN[8] methods are much worse than the NCSR [3] method. Compared with the NCSR method, the average PSNR gain of the proposed method can be up to 0.46 dB, showing the effectiveness of the proposed sparse codes learning method. Parts of the reconstructed HR images are shown in Fig. 2

Table 1: Average PSNR and SSIM results of the test methods (LR images generated with bicubic resizing function)

| Images | Se5 | | | Set14 | | | BSD100 | | |
|---|---|---|---|---|---|---|---|---|---|
| Upscaling | ×2 | ×3 | ×4 | ×2 | ×3 | ×4 | ×2 | ×3 | ×4 |
| SCSR[2] | - | 31.42 0.8821 | - | - | 28.31 0.7954 | - | - | 26.54 0.7729 | - |
| KK[7] | 36.22 0.9514 | 32.29 0.9037 | 30.03 0.8544 | 32.12 0.9029 | 28.39 0.8135 | 27.15 0.7422 | 31.08 0.8834 | 28.15 0.7780 | 26.69 0.7017 |
| A+[6] | 36.55 0.9544 | 32.59 0.9088 | 30.29 0.8603 | 32.28 0.9056 | 29.13 0.8188 | 27.33 0.7491 | 31.21 0.8863 | 28.29 0.7835 | 26.82 0.7087 |
| SRCNN[8] | 36.66 0.9542 | 32.75 0.9090 | 30.49 0.8628 | 32.45 0.9067 | 29.30 0.8215 | 27.50 0.7513 | 31.36 **0.8879** | 28.41 0.7863 | 26.90 0.7103 |
| NCSR[3] | 36.68 0.9550 | 33.05 0.9149 | 30.77 0.8720 | 32.26 0.9058 | 29.30 0.8239 | 27.52 0.7563 | 31.14 0.8863 | 28.37 0.7872 | 26.91 0.7143 |
| Proposed | **36.99** **0.9551** | **33.39** **0.9173** | **31.04** **0.8779** | **32.61** **0.9072** | **29.59** **0.8264** | **27.77** **0.7620** | **31.42** **0.8879** | **28.56** **0.7899** | **27.08** **0.7187** |

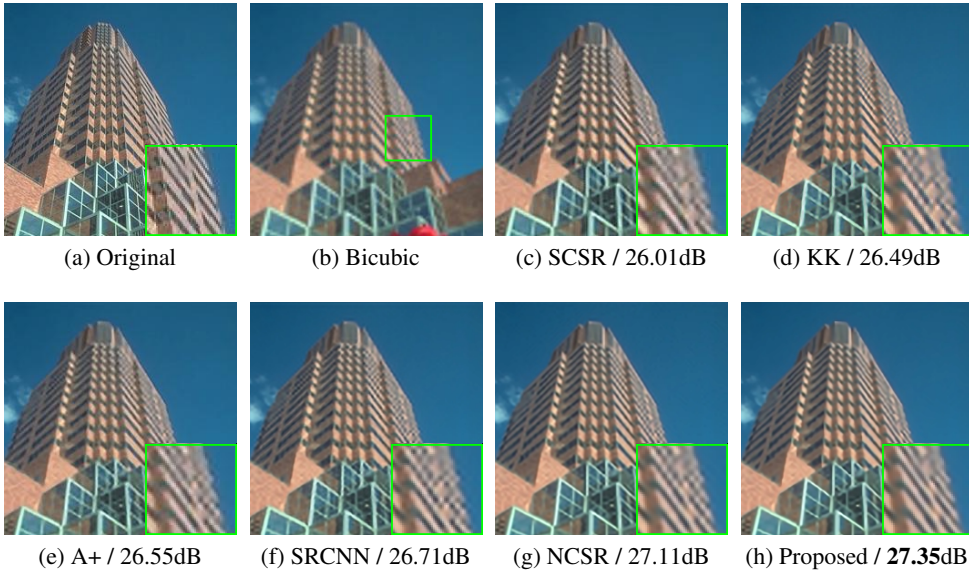

(a) Original    (b) Bicubic    (c) SCSR / 26.01dB    (d) KK / 26.49dB

(e) A+ / 26.55dB    (f) SRCNN / 26.71dB    (g) NCSR / 27.11dB    (h) Proposed / **27.35**dB

Figure 1: SR results on image '86000' of BSD100 of scaling factor 3 (LR image generated with bicubic interpolation function).

and Fig. 3. Obviously, the proposed method can recover sharper edges and finer details than other competing methods.

# 6 Conclusion

In this paper, we propose a novel approach for learning parametric sparse models for image super-resolution. Specifically, mapping functions between the LR patch and the sparse codes of the desirable HR patches are learned from a training set. Then, parametric sparse distributions are estimated from the learned sparse codes and those estimated from the input LR image. With the learned sparse models, the sparse codes and thus the HR image patches can be accurately recovered by solving a sparse coding problem. Experimental results show that the proposed SR method outperforms existing state-of-the-art methods in terms of both subjective and objective image qualities.

**Acknowledgments**

This work was supported in part by the Natural Science Foundation (NSF) of China under Grants(No. No. 61622210, 61471281, 61632019, 61472301, and 61390512), in part by the Specialized Research Fund for the Doctoral Program of Higher Education (No. 20130203130001).

Table 2: Average PSNR and SSIM results of the test methods of scaling factor 3 (LR images generated with Gaussian kernel followed by downsampling)

|  | SCSR[2] | KK[7] | A+[6] | SRCNN[8] | NCSR[3] | Proposed |
|---|---|---|---|---|---|---|
| Set5 | 30.22 | 30.28 | 29.39 | 30.20 | 33.03 | **33.49** |
|  | 0.8484 | 0.8536 | 0.8502 | 08514 | 0.9106 | **0.9165** |
| Set14 | 27.51 | 27.46 | 26.96 | 27.48 | 29.28 | **29.63** |
|  | 0.7619 | 0.7640 | 0.7627 | 0.7638 | 0.8203 | **0.8255** |
| BSD100 | 27.10 | 27.10 | 26.59 | 27.11 | 28.35 | **28.60** |
|  | 0.7338 | 0.7342 | 0.7331 | 0.7338 | 0.7841 | **0.7887** |

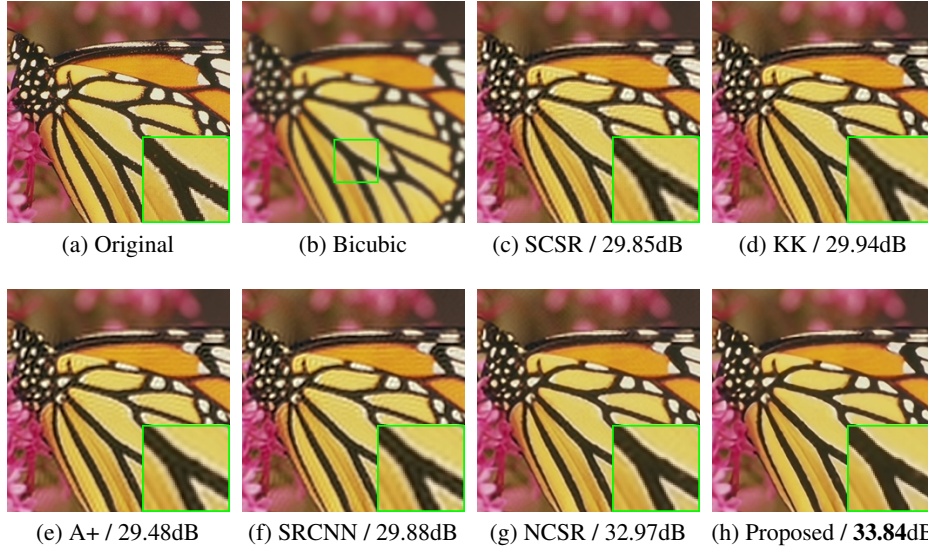

(a) Original    (b) Bicubic    (c) SCSR / 29.85dB    (d) KK / 29.94dB

(e) A+ / 29.48dB    (f) SRCNN / 29.88dB    (g) NCSR / 32.97dB    (h) Proposed / **33.84**dB

Figure 2: SR results on 'Monarch' from Set14 of scaling factor 3 (LR images generated with Gaussian blur followed downsampling).

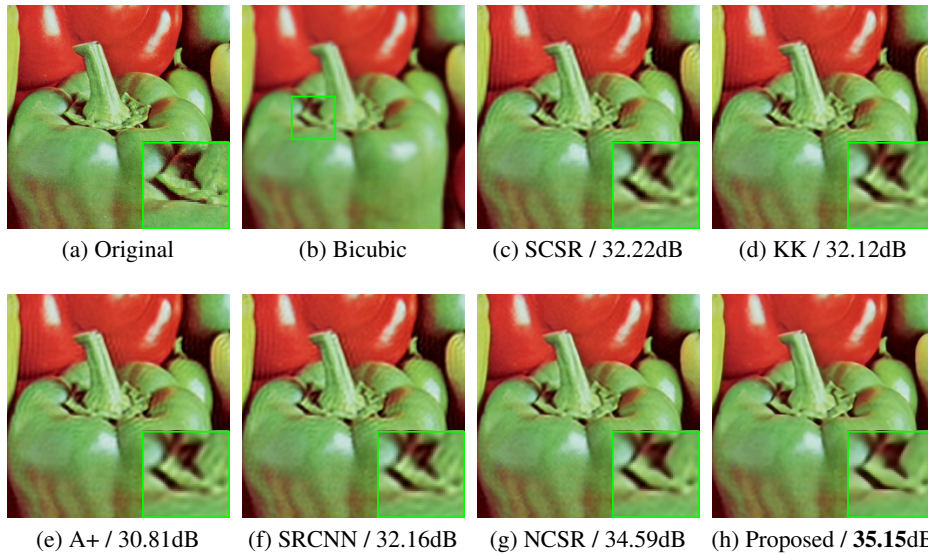

(a) Original    (b) Bicubic    (c) SCSR / 32.22dB    (d) KK / 32.12dB

(e) A+ / 30.81dB    (f) SRCNN / 32.16dB    (g) NCSR / 34.59dB    (h) Proposed / **35.15**dB

Figure 3: SR results on 'Pepper' from Set14 of scaling factor 3 (LR images generated with Gaussian blur followed downsampling).

# References

[1] A. Marquina and S. J. Osher. Image super-resolution by TV-regularization and bregman iteration. *Journal of Scientific Computing*, 37(3):367–382, 2008.

[2] J. Yang, J. Wright, T. S. Huang, and Y. Ma. Image super-resolution via sparse representation. *IEEE transactions on image processing*, 19(11):2861–2873, 2010.

[3] W. Dong, L. Zhang, G. Shi, and X. Li. Nonlocally centralized sparse representation for image restoration. *IEEE Transactions on Image Processing*, 22(4):1620–1630, 2013.

[4] W. Dong, G. Shi, Y. Ma, and X. Li. Image restoration via simultaneous sparse coding: Where structured sparsity meets gaussian scale mixture. *International Journal of Computer Vision*, 114(2-3):217–232, 2015.

[5] R. Timofte, V. De Smet, and L. Van Gool. Anchored neighborhood regression for fast example-based super-resolution. In *Proceedings of the IEEE International Conference on Computer Vision*, pages 1920–1927, 2013.

[6] R. Timofte, V. De Smet, and L. Van Gool. A+: Adjusted anchored neighborhood regression for fast super-resolution. In *Asian Conference on Computer Vision*, pages 111–126. Springer, 2014.

[7] K. I. Kim and Y. Kwon. Single-image super-resolution using sparse regression and natural image prior. *IEEE Transactions on Pattern Analysis and Machine Intelligence*, 32(6):1127–1133, 2010.

[8] C. Dong, C. C. Loy, K. He, and X. Tang. Image super-resolution using deep convolutional networks. *IEEE transactions on pattern analysis and machine intelligence*, 38(2):295–307, 2016.

[9] M. Bevilacqua, A. Roumy, C. Guillemot, and M. L. AlberiMorel. Low-complexity single-image super-resolution based on nonnegative neighbor embedding. 2012.

[10] R. Zeyde, M. Elad, and M. Protter. On single image scale-up using sparse-representations. In *International conference on curves and surfaces*, pages 711–730. Springer, 2010.

[11] D. Martin, C. Fowlkes, D. Tal, and J. Malik. A database of human segmented natural images and its application to evaluating segmentation algorithms and measuring ecological statistics. In *Computer Vision, 2001. ICCV 2001. Proceedings. Eighth IEEE International Conference on*, volume 2, pages 416–423. IEEE, 2001.

[12] Y. Li, W. Dong, G. Shi, and X. Xie. Learning parametric distributions for image super-resolution: Where patch matching meets sparse coding. In *Proceedings of the IEEE International Conference on Computer Vision*, pages 450–458, 2015.

[13] W. Dong, L. Zhang, G. Shi, and X. Wu. Image deblurring and super-resolution by adaptive sparse domain selection and adaptive regularization. *IEEE Transactions on Image Processing*, 20(7):1838–1857, 2011.

[14] W. Dong, L. Zhang, and G. Shi. Centralized sparse representation for image restoration. In *2011 International Conference on Computer Vision*, pages 1259–1266. IEEE, 2011.

[15] G. Yu, G. Sapiro, and S. Mallat. Solving inverse problems with piecewise linear estimators: From gaussian mixture models to structured sparsity. *IEEE Transactions on Image Processing*, 21(5):2481–2499, 2012.

[16] A. Krizhevsky, I. Sutskever, and G. E. Hinton. Imagenet classification with deep convolutional neural networks. In *Advances in neural information processing systems*, pages 1097–1105, 2012.

[17] M. Irani and S. Peleg. Motion analysis for image enhancement: Resolution, occlusion, and transparency. *Journal of Visual Communication and Image Representation*, 4(4):324–335, 1993.

[18] D. Dai, R. Timofte, and L. Van Gool. Jointly optimized regressors for image super-resolution. In *Computer Graphics Forum*, volume 34, pages 95–104. Wiley Online Library, 2015.

[19] K. Egiazarian and V. Katkovnik. Single image super-resolution via BM3D sparse coding. In *Signal Processing Conference (EUSIPCO), 2015 23rd European*, pages 2849–2853. IEEE, 2015.

